# Silicon Models
# for
# Auditory Scene Analysis

**John Lazzaro and John Wawrzynek**
CS Division
UC Berkeley
Berkeley, CA 94720-1776
`lazzaro@cs.berkeley.edu, johnw@cs.berkeley.edu`

## Abstract

We are developing special-purpose, low-power analog-to-digital converters for speech and music applications, that feature analog circuit models of biological audition to process the audio signal before conversion. This paper describes our most recent converter design, and a working system that uses several copies of the chip to compute multiple representations of sound from an analog input. This multi-representation system demonstrates the plausibility of inexpensively implementing an auditory scene analysis approach to sound processing.

## 1. INTRODUCTION

The visual system computes multiple representations of the retinal image, such as motion, orientation, and stereopsis, as an early step in scene analysis. Likewise, the auditory brainstem computes secondary representations of sound, emphasizing properties such as binaural disparity, periodicity, and temporal onsets. Recent research in auditory scene analysis involves using computational models of these auditory brainstem representations in engineering applications.

Computation is a major limitation in auditory scene analysis research: the complete auditory processing system described in (Brown and Cooke, 1994) operates at approximately 4000 times real time, running under UNIX on a Sun SPARCstation 1. Standard approaches to hardware acceleration for signal processing algorithms could be used to ease this computational burden in a research environment; a variety of parallel, fixed-point hardware products would work well on these algorithms.

However, hardware solutions appropriate for a research environment may not be well suited for accelerating algorithms in cost-sensitive, battery-operated consumer products. Possible product applications of auditory algorithms include robust pitch-tracking systems for musical instrument applications, and small-vocabulary, speaker-independent wordspotting systems for control applications.

In these applications, the input takes an analog form: a voltage signal from a microphone or a guitar pickup. Low-power analog circuits that compute auditory representations have been implemented and characterized by several research groups – these working research prototypes include several generation of cochlear models (Lyon and Mead, 1988), periodicity models, and binaural models. These circuits could be used to compute auditory representations directly on the analog signal, in real-time, using these low-power, area-efficient analog circuits.

Using analog computation successfully in a system presents many practical difficulties; the density and power advantages of the analog approach are often lost in the process of system integration. One successful IC architecture that uses analog computation in a system is the special-purpose analog to digital converter, that includes analog, non-linear pre-processing before or during data conversion. For example, converters that include logarithmic waveform compression before digitization are commercially viable components.

Using this component type as a model, we have been developing special-purpose, low-power analog-to-digital converters for speech and audio applications; this paper describes our most recent converter design, and a working system that uses several copies of the chip to compute multiple representations of sound.

## 2. CONVERTER DESIGN

Figure 1 shows an architectural block diagram of our current converter design. The 35,000 transistor chip was fabricated in the $2\mu$m, n-well process of Orbit Semiconductor, brokered through MOSIS; the circuit is fully functional. Below is a summary of the general architectural features of this chip; unless otherwise referenced, circuit details are similar to the converter design described in (Lazzaro *et al.*, 1994).

• An analog audio signal serves as input to the chip; dynamic range is 40dB to 60dB (1-10mV to 1V peak, dependent on measurement criteria).

• This signal is processed by analog circuits that model cochlear processing (Lyon and Mead, 1988) and sensory transduction; the audio signal is transformed into 119 wavelet-filtered, half-wave rectified, non-linearly compressed audio signals. The cycle-by-cycle waveform of each signal is preserved; no temporal smoothing is performed.

• Two additional analog processing blocks follow this initial cochlear processing, a temporal autocorrelation processor and a temporal adaptation processor. Each block transforms the input array into a new representation of equal size; alternatively, the block can be programmed to pass its input vector to its output without alteration.

• The output circuits of the final processing block are pulse generators, which code the signal as a pattern of fixed-width, fixed-height spikes. All the information in the representation is contained in the onset times of the pulses.

• The activity on this array is sent off-chip via an asynchronous parallel bus. The converter chip acts as a sender on the bus; a digital host processor is the receiver. The converter initiates a transaction on the bus to communicate the onset of a pulse in the array; the data value on the bus is a number indicating which unit in the array pulsed. The time of transaction initiation carries essential information. This coding method is also known as the address-event representation.

• Many converters can be used in the same system, sharing the same asynchronous output bus (Lazzaro and Wawrzynek, 1995). No extra components are needed to implement bus sharing; the converter bus design includes extra signals and logic that implements multi-chip bus arbitration. This feature is a major difference between this design and (Lazzaro *et al.*, 1994).

• The converter includes a digitally-controllable parameter storage and generation system; 25 tunable parameters control the behavior of the analog processing blocks. Programmability supports the creation of multi-converter systems that use a single chip design: each chip receives the same analog signal, but processes the signal in different ways, as determined by the parameter values for each chip.

• Non-volatile analog storage elements are used to store the parameters; parameters are changeable via Fowler-Nordhiem tunneling, using a 5V control input bus. Many converters can share the same control bus. Parameter values can be sensed by activating a control mode, which sends parameter information on the converter output bus. Apart from two high-voltage power supply pins, and a trimming input pin for tunneling pulse width, all control voltages used in this converter are generated on-chip.

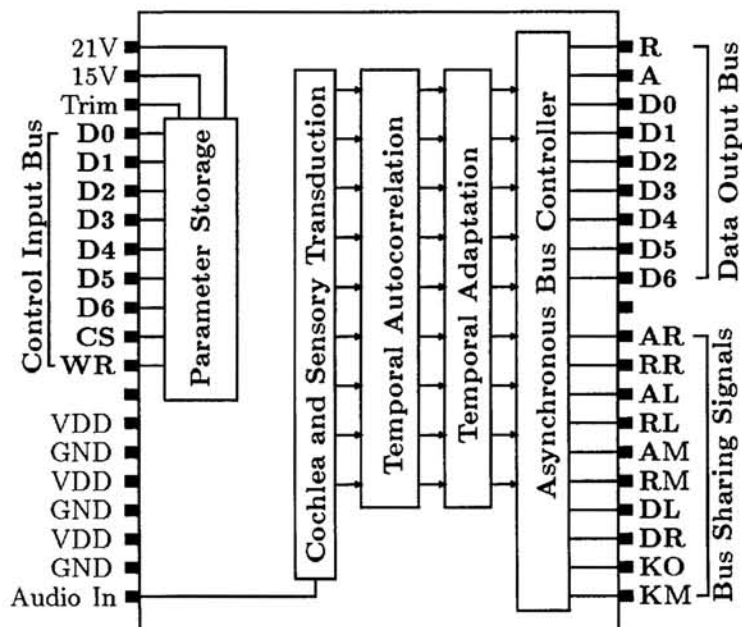

**Figure 1.** Block diagram of the converter chip. Most of the 40 pins of the chip are dedicated to the data output and control input buses, and to the control signals for coordinating bus sharing in multi-converter systems.

## 3. SYSTEM DESIGN

Figure 2 shows a block diagram of a system that uses three copies of the converter chip to compute multiple representations of sound; the system acts as a real-time audio input device to a Sun workstation. An analog audio input connects to each converter; this input can be from a pre-amplified microphone, for spontaneous input, or from the analog audio signal of the workstation, for controlled experiments.

The asynchronous output buses from the three chips are connected together, to produce a single output address space for the system; no external components are needed for output bus sharing and arbitration. The onset time of a transaction carries essential information on this bus; additional logic on this board adds a 16-bit timestamp to each bus transaction, coding the onset time with $20\mu s$ resolution. The control input buses for the three chips are also connected together to produce a single input address space, using external logic for address decoding. We use a commercial interface board to link the workstation with these system buses.

## 4. SYSTEM PERFORMANCE

We designed a software environment, **Aer**, to support real-time, low-latency data visualization of the multi-converter system. Using Aer, we can easily experiment with different converter tunings. Figure 3 shows a screen from Aer, showing data from the three converters as a function of time; the input sound for this screen is a short 800 Hz tone burst, followed by a sinusoid sweep from 300 Hz to 3 Khz. The top ("Spectral Shape") and bottom ("Onset") representations are raw data from converters 1 and 3, as marked on Figure 2, tuned for different responses. The output channel number is plotted vertically; each dot represents a pulse.

The top representation codes for periodicity-based spectral shape; for this representation, the temporal autocorrelation block (see Figure 1) is activated, and the temporal adaptation block is inactivated. Spectral frequency is mapped logarithmically on the vertical dimension, from 300 Hz to 4 Khz; the activity in each channel is the periodic waveform present at that frequency. The difference between a periodicity-based spectral method and a resonant spectral method can be seen in the response to the 800 Hz sinusoid onset: the periodicity representation shows activity only around the 800 Hz channels, whereas a spectral representation would show broadband transient activity at tone onset.

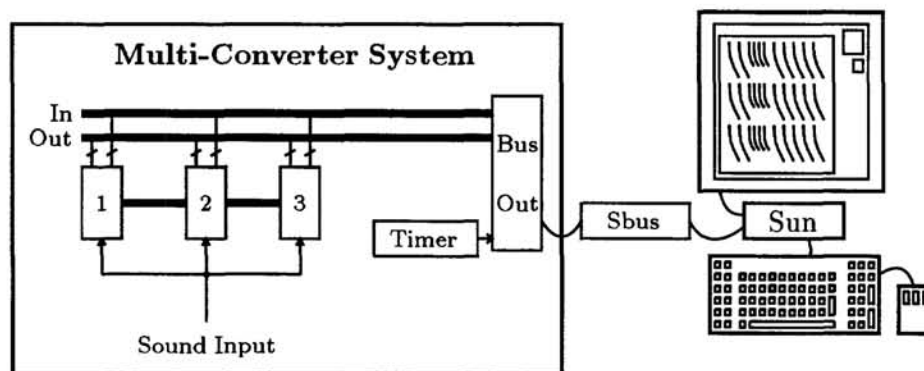

**Figure 2.** Block diagram of the multi-converter system.

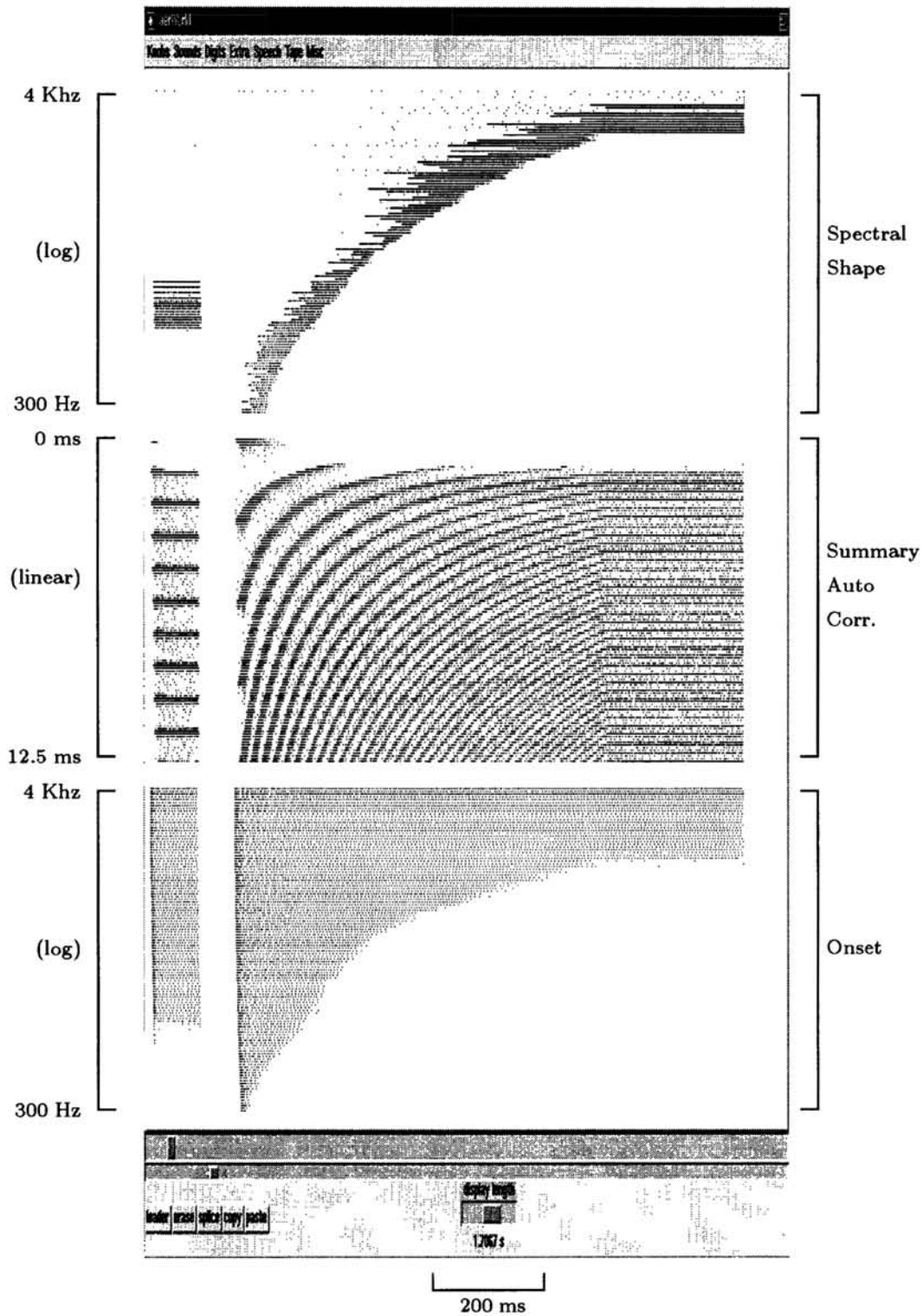

**Figure 3.** Data from the multi-converter system, in response to a 800-Hz pure tone, followed by a sinusoidal sweep from 300Hz to 3Khz.

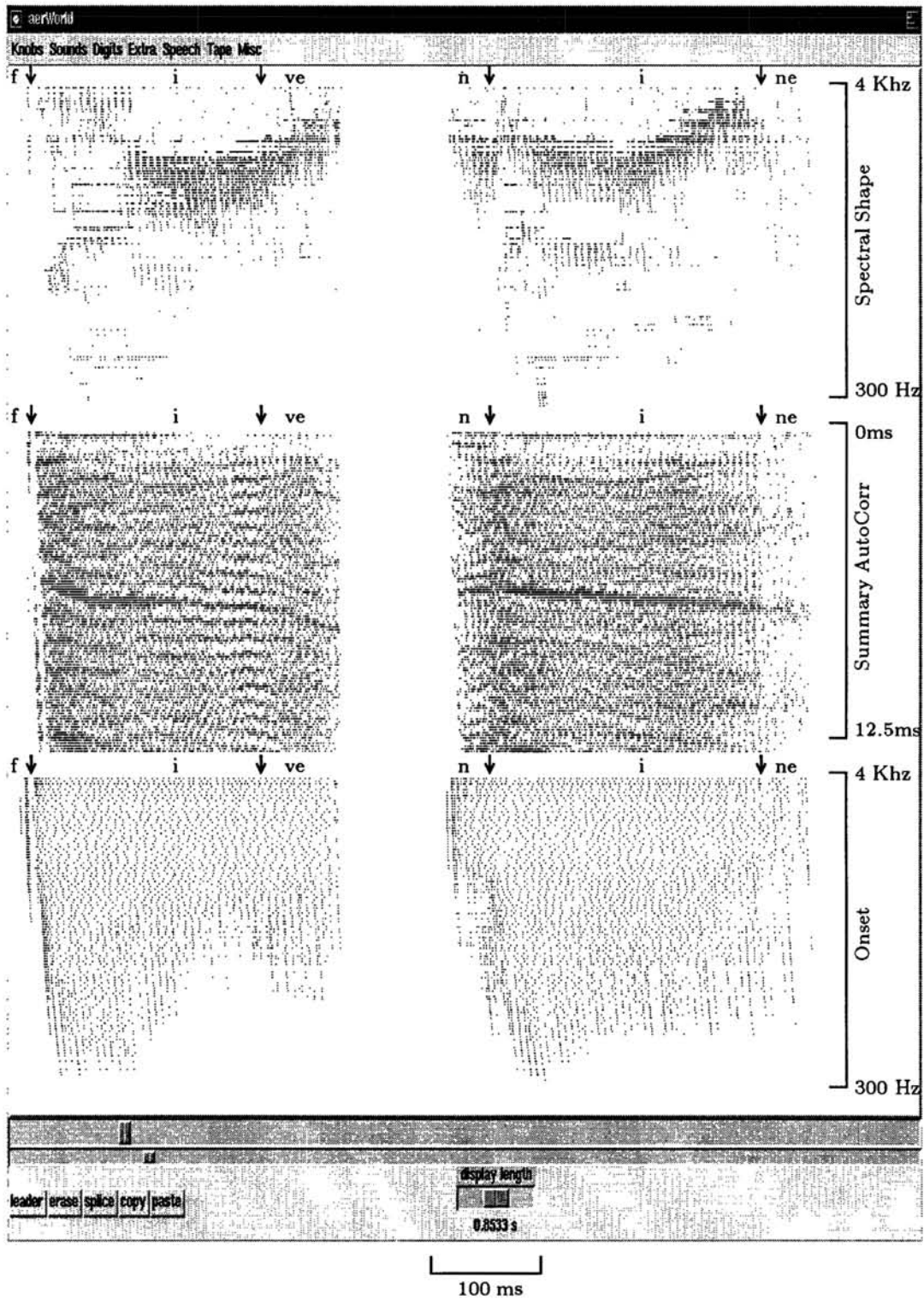

**Figure 4.** Data from the multi-converter system, in response to the word "five" followed by the word "nine".

The bottom representation codes for temporal onsets; for this representation, the temporal adaptation block is activated, and the temporal autocorrelation block is inactivated. The spectral filtering of the representation reflects the silicon cochlea tuning: a low-pass response with a sharp cutoff and a small resonant peak at the best frequency of the filter. The black, wideband lines at the start of the 800 Hz tone and the sinusoid sweep illustrate the temporal adaptation.

The middle ("Summary Auto Corr.") representation is a summary autocorrelogram, useful for pitch processing and voiced/unvoiced decisions in speech recognition. This representation is not raw data from a converter; software post-processing is performed on the converter output to produce the final result. The frequency response of converter 2 is set as in the bottom representation; the temporal adaptation response, however, is set to a 100 millisecond time constant. The converter output pulse rates are set so that the cycle-by-cycle waveform information for each output channel is preserved in the output.

To complete the representation, a set of running autocorrelation functions $x(t)x(t-\tau)$ is computed for $\tau = k\,105\mu s, k = 1\ldots120$, for each of the 119 output channels. These autocorrelation functions are summed over all output channels to produce the final representation; $\tau$ is plotted as a linear function of time on the vertical axis. The correlation multiplication can be efficiently implemented by integer subtraction and comparison of pulse timestamps; the summation over channels is simply the merging of lists of bus transactions. The middle representation in Figure 3 shows the qualitative characteristics of the summary autocorrelogram: a repetitive band structure in response to periodic sounds.

Figure 4 shows the output response of the multi-converter system in response to telephone-bandwidth-limited speech; the phonetic boundaries of the two words, "five" and "nine", are marked by arrows. The vowel formant information is shown most clearly by the strong peaks in the spectral shape representation; the wideband information in the "f" of five is easily seen in the onset representation. The summary autocorrelation representation shows a clear texture break between vowels and the voiced "n" and "v" sounds.

## Acknowledgements

Thanks to Richard Lyon and Peter Cariani for summary autocorrelogram discussions. Funded by the Office of Naval Research (URI-N00014-92-J-1672).

## References

Brown, G.J. and Cooke, M. (1994). Computational auditory scene analysis. *Computer Speech and Language*, **8:4**, pp. 297-336.

Lazzaro, J. P. and Wawrzynek, J. (1995). A multi-sender asynchronous extension to the address-event protocol. In Dally, W. J., Poulton, J. W., Ishii, A. T. (eds), *16th Conference on Advanced Research in VLSI*, pp. 158–169.

Lazzaro, J. P., Wawrzynek, J., and Kramer, A (1994). Systems technologies for silicon auditory models. *IEEE Micro*, **14:3**. 7-15.

Lyon, R. F., and Mead, C. (1988). An analog electronic cochlea. *IEEE Trans. Acoust., Speech, Signal Processing* vol. 36, pp. 1119-1134.